# Global Solution of Fully-Observed Variational Bayesian Matrix Factorization is Column-Wise Independent

**Shinichi Nakajima**
Nikon Corporation
Tokyo, 140-8601, Japan
nakajima.s@nikon.co.jp

**Masashi Sugiyama**
Tokyo Institute of Technology
Tokyo 152-8552, Japan
sugi@cs.titech.ac.jp

**Derin Babacan**
University of Illinois at Urbana-Champaign
Urbana, IL 61801, USA
dbabacan@illinois.edu

## Abstract

Variational Bayesian matrix factorization (VBMF) efficiently approximates the posterior distribution of factorized matrices by assuming *matrix-wise independence* of the two factors. A recent study on fully-observed VBMF showed that, under a stronger assumption that the two factorized matrices are *column-wise* independent, the global optimal solution can be analytically computed. However, it was not clear how restrictive the column-wise independence assumption is. In this paper, we prove that the global solution under matrix-wise independence is actually column-wise independent, implying that the column-wise independence assumption is harmless. A practical consequence of our theoretical finding is that the global solution under matrix-wise independence (which is a standard setup) can be obtained analytically in a computationally very efficient way without any iterative algorithms. We experimentally illustrate advantages of using our analytic solution in probabilistic principal component analysis.

## 1 Introduction

The goal of *matrix factorization* (MF) is to approximate an observed matrix by a low-rank one. In this paper, we consider *fully-observed MF* where the observed matrix has no missing entry[1]. This formulation includes classical multivariate analysis techniques based on singular-value decomposition such as *principal component analysis (PCA)* [9] and *canonical correlation analysis* [10].

In the framework of *probabilistic MF* [20, 17, 19], posterior distributions of factorized matrices are considered. Since exact inference is computationally intractable, the Laplace approximation [3], the Markov chain Monte Carlo sampling [3, 18], and the variational Bayesian (VB) approximation [4, 13, 16, 15] were used for approximate inference in practice. Among them, the VB approximation seems to be a popular choice due to its high accuracy and computational efficiency.

In the original VBMF [4, 13], factored matrices are assumed to be *matrix-wise* independent, and a local optimal solution is computed by an iterative algorithm. A simplified variant of VBMF (simpleVBMF) was also proposed [16], which assumes a stronger constraint that the factored matrices

are *column-wise* independent. A notable advantage of simpleVBMF is that the global optimal solution can be computed *analytically* in a computationally very efficient way [15].

Intuitively, it is suspected that simpleVBMF only possesses weaker approximation ability due to its stronger column-wise independence assumption. However, it was reported that no clear performance degradation was observed in experiments [14]. Thus, simpleVBMF would be a practically useful approach. Nevertheless, the influence of the stronger column-wise independence assumption was not elucidated beyond this empirical evaluation.

The main contribution of this paper is to theoretically show that the column-wise independence assumption does *not* degrade the performance. More specifically, we prove that a global optimal solution of the original VBMF is actually column-wise independent. Thus, a global optimal solution of the original VBMF can be obtained by the analytic-form solution of simpleVBMF—no computationally-expensive iterative algorithm is necessary. We show the usefulness of the analytic-form solution through experiments on probabilistic PCA.

## 2  Formulation

In this section, we first formulate the problem of probabilistic MF, and then introduce the VB approximation and its simplified variant.

### 2.1  Probabilistic Matrix Factorization

The probabilistic MF model is given as follows [19]:

$$p(Y|A, B) \propto \exp\left(-\frac{1}{2\sigma^2}\|Y - BA^\top\|_{\text{Fro}}^2\right), \tag{1}$$

$$p(A) \propto \exp\left(-\frac{1}{2}\text{tr}\left(AC_A^{-1}A^\top\right)\right), \qquad p(B) \propto \exp\left(-\frac{1}{2}\text{tr}\left(BC_B^{-1}B^\top\right)\right), \tag{2}$$

where $Y \in \mathbb{R}^{L \times M}$ is an observed matrix, $A \in \mathbb{R}^{M \times H}$ and $B \in \mathbb{R}^{L \times H}$ are parameter matrices to be estimated, and $\sigma^2$ is the noise variance. Here, we denote by $\top$ the transpose of a matrix or vector, by $\|\cdot\|_{\text{Fro}}$ the Frobenius norm, and by $\text{tr}(\cdot)$ the trace of a matrix. We assume that the prior covariance matrices $C_A$ and $C_B$ are diagonal and positive definite, i.e.,

$$C_A = \text{diag}(c_{a_1}^2, \ldots, c_{a_H}^2), \quad C_B = \text{diag}(c_{b_1}^2, \ldots, c_{b_H}^2) \qquad \text{for } c_{a_h}, c_{b_h} > 0, h = 1, \ldots, H.$$

Without loss of generality, we assume that the diagonal entries of the product $C_A C_B$ are arranged in the non-increasing order, i.e., $c_{a_h} c_{b_h} \geq c_{a_{h'}} c_{b_{h'}}$ for any pair $h < h'$.

Throughout the paper, we denote a column vector of a matrix by a bold smaller letter, and a row vector by a bold smaller letter with a tilde, namely,

$$A = (\boldsymbol{a}_1, \ldots, \boldsymbol{a}_H) = (\widetilde{\boldsymbol{a}}_1, \ldots, \widetilde{\boldsymbol{a}}_M)^\top \in \mathbb{R}^{M \times H}, \quad B = (\boldsymbol{b}_1, \ldots, \boldsymbol{b}_H) = \left(\widetilde{\boldsymbol{b}}_1, \ldots, \widetilde{\boldsymbol{b}}_L\right)^\top \in \mathbb{R}^{L \times H}.$$

### 2.2  Variational Bayesian Approximation

The Bayes posterior is written as

$$p(A, B|Y) = \frac{p(Y|A, B)p(A)p(B)}{Z(Y)}, \tag{3}$$

where $Z(Y) = \langle p(Y|A, B)\rangle_{p(A)p(B)}$ is the marginal likelihood. Here, $\langle\cdot\rangle_p$ denotes the expectation over the distribution $p$. Since the Bayes posterior (3) is computationally intractable, the VB approximation was proposed [4, 13, 16, 15].

Let $r(A, B)$, or $r$ for short, be a trial distribution. The following functional with respect to $r$ is called the free energy:

$$F(r|Y) = \left\langle \log \frac{r(A, B)}{p(Y|A, B)p(A)p(B)} \right\rangle_{r(A,B)} = \left\langle \log \frac{r(A, B)}{p(A, B|Y)} \right\rangle_{r(A,B)} - \log Z(Y). \tag{4}$$

In the last equation, the first term is the Kullback-Leibler (KL) distance from the trial distribution to the Bayes posterior, and the second term is a constant. Therefore, minimizing the free energy (4) amounts to finding the distribution closest to the Bayes posterior in the sense of the KL distance. In the VB approximation, the free energy (4) is minimized over some restricted function space.

A standard constraint for the MF model is *matrix-wise* independence [4, 13], i.e.,

$$r^{\text{VB}}(A, B) = r_{\text{A}}^{\text{VB}}(A) r_{\text{B}}^{\text{VB}}(B). \tag{5}$$

This constraint breaks off the entanglement between the parameter matrices $A$ and $B$, and leads to a computationally-tractable iterative algorithm. Using the variational method, we can show that, under the constraint (5), the VB posterior minimizing the free energy (4) is written as

$$r^{\text{VB}}(A, B) = \prod_{m=1}^{M} \mathcal{N}_H(\widetilde{\boldsymbol{a}}_m; \widetilde{\widehat{\boldsymbol{a}}}_m, \Sigma_A) \prod_{l=1}^{L} \mathcal{N}_H(\widetilde{\boldsymbol{b}}_l; \widetilde{\widehat{\boldsymbol{b}}}_l, \Sigma_B),$$

where the parameters satisfy

$$\widehat{A} = \left(\widetilde{\widehat{\boldsymbol{a}}}_1, \dots, \widetilde{\widehat{\boldsymbol{a}}}_M\right)^{\top} = Y^{\top} \widehat{B} \frac{\Sigma_A}{\sigma^2}, \qquad \Sigma_A = \sigma^2 \left(\widehat{B}^{\top} \widehat{B} + L\Sigma_B + \sigma^2 C_A^{-1}\right)^{-1}, \tag{6}$$

$$\widehat{B} = \left(\widetilde{\widehat{\boldsymbol{b}}}_1, \dots, \widetilde{\widehat{\boldsymbol{b}}}_L\right)^{\top} = Y \widehat{A} \frac{\Sigma_B}{\sigma^2}, \qquad \Sigma_B = \sigma^2 \left(\widehat{A}^{\top} \widehat{A} + M\Sigma_A + \sigma^2 C_B^{-1}\right)^{-1}. \tag{7}$$

Here, $\mathcal{N}_d(\cdot; \boldsymbol{\mu}, \Sigma)$ denotes the $d$-dimensional Gaussian distribution with mean $\boldsymbol{\mu}$ and covariance matrix $\Sigma$. Iteratively updating the parameters $\widehat{A}, \Sigma_A, \widehat{B}$, and $\Sigma_B$ by Eqs.(6) and (7) until convergence gives a local minimum of the free energy (4).

When the noise variance $\sigma^2$ is unknown, it can also be estimated based on the free energy minimization. The update rule for $\sigma^2$ is given by

$$\sigma^2 = \frac{\|Y\|_{\text{Fro}}^2 - \text{tr}(2Y^{\top} \widehat{B} \widehat{A}^{\top}) + \text{tr}\left((\widehat{A}^{\top} \widehat{A} + M\Sigma_A)(\widehat{B}^{\top} \widehat{B} + L\Sigma_B)\right)}{LM}. \tag{8}$$

Furthermore, in the *empirical* Bayesian scenario, the hyperparameters $C_A$ and $C_B$ are also estimated from data. In this scenario, $C_A$ and $C_B$ are updated in each iteration by the following formulas:

$$c_{a_h}^2 = \|\widehat{\boldsymbol{a}}_h\|^2/M + (\Sigma_A)_{hh}, \qquad c_{b_h}^2 = \|\widehat{\boldsymbol{b}}_h\|^2/L + (\Sigma_B)_{hh}. \tag{9}$$

## 2.3 SimpleVB Approximation

A simplified variant, called the simpleVB approximation, assumes *column-wise* independence of each matrix [16, 15], i.e.,

$$r^{\text{simpleVB}}(A, B) = \prod_{h=1}^{H} r_{a_h}^{\text{simpleVB}}(\boldsymbol{a}_h) \prod_{h=1}^{H} r_{b_h}^{\text{simpleVB}}(\boldsymbol{b}_h). \tag{10}$$

This constraint restricts the covariances $\Sigma_A$ and $\Sigma_B$ to be diagonal, and thus necessary memory storage and computational cost are substantially reduced [16]. The simpleVB posterior can be written as

$$r^{\text{simpleVB}}(A, B) = \prod_{h=1}^{H} \mathcal{N}_M(\boldsymbol{a}_h; \widehat{\boldsymbol{a}}_h, \sigma_{a_h}^2 I_M) \mathcal{N}_L(\boldsymbol{b}_h; \widehat{\boldsymbol{b}}_h, \sigma_{b_h}^2 I_L),$$

where the parameters satisfy

$$\widehat{\boldsymbol{a}}_h = \frac{\sigma_{a_h}^2}{\sigma^2} \left(Y - \sum_{h' \neq h} \widehat{\boldsymbol{b}}_{h'} \widehat{\boldsymbol{a}}_{h'}^{\top}\right)^{\top} \widehat{\boldsymbol{b}}_h, \qquad \sigma_{a_h}^2 = \sigma^2 \left(\|\widehat{\boldsymbol{b}}_h\|^2 + L\sigma_{b_h}^2 + \sigma^2 c_{a_h}^{-2}\right)^{-1}, \tag{11}$$

$$\widehat{\boldsymbol{b}}_h = \frac{\sigma_{b_h}^2}{\sigma^2} \left(Y - \sum_{h' \neq h} \widehat{\boldsymbol{b}}_{h'} \widehat{\boldsymbol{a}}_{h'}^{\top}\right) \widehat{\boldsymbol{a}}_h, \qquad \sigma_{b_h}^2 = \sigma^2 \left(\|\widehat{\boldsymbol{a}}_h\|^2 + M\sigma_{a_h}^2 + \sigma^2 c_{b_h}^{-2}\right)^{-1}. \tag{12}$$

Here, $I_d$ denotes the $d$-dimensional identity matrix. Iterating Eqs.(11) and (12) until convergence, we can obtain a local minimum of the free energy. Eqs.(8) and (9) are similarly applied if the noise variance $\sigma^2$ is unknown and in the empirical Bayesian scenario, respectively.

A recent study has derived the analytic solution for simpleVB when the observed matrix has no missing entry [15]. This work made simpleVB more attractive, because it did not only provide substantial reduction of computation costs, but also guaranteed the global optimality of the solution. However, it was not clear how restrictive the column-wise independence assumption is, beyond its experimental success [14]. In the next section, we theoretically show that the column-wise independence assumption is actually harmless.

## 3 Analytic Solution of VBMF under *Matrix-wise* Independence

Under the *matrix-wise* independence constraint (5), the free energy (4) can be written as

$$F = \langle \log r(A) + \log r(B) - \log p(Y|A, B)p(A)p(B) \rangle_{r(A)r(B)}$$

$$= \frac{LM}{2} \log \sigma^2 + \frac{M}{2} \log \frac{|C_A|}{|\Sigma_A|} + \frac{L}{2} \log \frac{|C_B|}{|\Sigma_B|} + \frac{\|Y\|^2}{2\sigma^2} + \text{const.}$$

$$+ \frac{1}{2} \text{tr} \left\{ C_A^{-1} \left( \widehat{A}^\top \widehat{A} + M\Sigma_A \right) + C_B^{-1} \left( \widehat{B}^\top \widehat{B} + L\Sigma_B \right) \right.$$

$$\left. + \sigma^{-2} \left( -2\widehat{A}^\top Y^\top \widehat{B} + \left( \widehat{A}^\top \widehat{A} + M\Sigma_A \right) \left( \widehat{B}^\top \widehat{B} + L\Sigma_B \right) \right) \right\}. \quad (13)$$

Note that Eqs.(6) and (7) together form the stationarity condition of Eq.(13) with respect to $\widehat{A}$, $\widehat{B}$, $\Sigma_A$, and $\Sigma_B$.

Below, we show that a global solution of $\Sigma_A$ and $\Sigma_B$ is diagonal. When the product $C_A C_B$ is non-degenerate (i.e., $c_{a_h} c_{b_h} > c_{a_{h'}} c_{b_{h'}}$ for any pair $h < h'$), the global solution is unique and diagonal. On the other hand, when $C_A C_B$ is degenerate, the global solutions are not unique because arbitrary rotation in the degenerate subspace is possible without changing the free energy. However, still one of the equivalent solutions is always diagonal.

**Theorem 1** *Diagonal $\Sigma_A$ and $\Sigma_B$ minimize the free energy* (13).

The basic idea of our proof is that, since minimizing the free energy (13) with respect to $A$, $B$, $\Sigma_A$, and $\Sigma_B$ is too complicated, we focus on a restricted space written in a particular form that includes the optimal solution. From necessary conditions for optimality, we can deduce that the solutions $\Sigma_A$ and $\Sigma_B$ are diagonal.

Below, we describe the outline of the proof for non-degenerate $C_A C_B$. The complete proof for general cases is omitted because of the page limit.

**(Sketch of proof of Theorem 1)** Assume that $(A^*, B^*, \Sigma_A^*, \Sigma_B^*)$ is a minimizer of the free energy (13), and consider the following set of parameters specified by an $H \times H$ orthogonal matrix $\Omega$:

$$\widehat{A} = A^* C_A^{-1/2} \Omega^\top C_A^{1/2}, \qquad \Sigma_A = C_A^{1/2} \Omega C_A^{-1/2} \Sigma_A^* C_A^{-1/2} \Omega^\top C_A^{1/2},$$

$$\widehat{B} = B^* C_A^{1/2} \Omega^\top C_A^{-1/2}, \qquad \Sigma_B = C_A^{-1/2} \Omega C_A^{1/2} \Sigma_B^* C_A^{1/2} \Omega^\top C_A^{-1/2}.$$

Note that $\widehat{B}\widehat{A}^\top$ is invariant with respect to $\Omega$, and $(\widehat{A}, \widehat{B}, \Sigma_A, \Sigma_B) = (A^*, B^*, \Sigma_A^*, \Sigma_B^*)$ holds if $\Omega = I_H$. Then, as a function of $\Omega$, the free energy (13) can be simplified as

$$F(\Omega) = \frac{1}{2} \text{tr} \left\{ C_A^{-1} C_B^{-1} \Omega C_A^{1/2} \left( B^{*\top} B^* + L\Sigma_B^* \right) C_A^{1/2} \Omega^\top \right\} + \text{const.}$$

This is necessarily minimized at $\Omega = I_H$, because we assumed that $(A^*, B^*, \Sigma_A^*, \Sigma_B^*)$ is a minimizer. We can show that $F(\Omega)$ is minimized at $\Omega = I_H$ only if $B^{*\top} B^* + L\Sigma_B^*$ is diagonal. This implies that $\Sigma_A^*$ (see Eq.(6)) should be diagonal.

Similarly, we consider another set of parameters specified by an $H \times H$ orthogonal matrix $\Omega'$:

$$\widehat{A} = A^* C_B^{1/2} \Omega'^\top C_B^{-1/2}, \qquad \Sigma_A = C_B^{-1/2} \Omega' C_B^{1/2} \Sigma_A^* C_B^{1/2} \Omega'^\top C_B^{-1/2},$$

$$\widehat{B} = B^* C_B^{-1/2} \Omega'^\top C_B^{1/2}, \qquad \Sigma_B = C_B^{1/2} \Omega' C_B^{-1/2} \Sigma_B^* C_B^{-1/2} \Omega'^\top C_B^{1/2}.$$

Then, as a function of $\Omega'$, the free energy (13) can be expressed as

$$F(\Omega') = \frac{1}{2}\mathrm{tr}\left\{C_A^{-1}C_B^{-1}\Omega'C_B^{1/2}\left(A^{*\top}A^* + M\Sigma_A^*\right)C_B^{1/2}\Omega'^\top\right\} + \text{const.}$$

Similarly, this is minimized at $\Omega' = I_H$ only if $A^{*\top}A^* + M\Sigma_A^*$ is diagonal. Thus, $\Sigma_B^*$ should be diagonal (see Eq.(7)). $\qquad\square$

The result that $\Sigma_A$ and $\Sigma_B$ become diagonal would be natural because we assumed the independent Gaussian prior on $A$ and $B$: the fact that any $Y$ can be decomposed into orthogonal components may imply that the observation $Y$ cannot convey any preference for singular-component-wise correlation. Note, however, that Theorem 1 does not necessarily hold when the observed matrix has missing entries.

Theorem 1 implies that the stronger *column-wise* independence constraint (10) does not degrade approximation accuracy, and the VB solution under *matrix-wise* independence (5) essentially agrees with the simpleVB solution. Consequently, we can obtain a global analytic solution for VB, by combining Theorem 1 above with Theorem 1 in [15]:

**Corollary 1** *Let $\gamma_h$ ($\geq 0$) be the $h$-th largest singular value of $Y$, and let $\boldsymbol{\omega}_{a_h}$ and $\boldsymbol{\omega}_{b_h}$ be the associated right and left singular vectors:*

$$Y = \sum_{h=1}^{L} \gamma_h \boldsymbol{\omega}_{b_h}\boldsymbol{\omega}_{a_h}^\top.$$

*Let $\widehat{\gamma}_h$ be the* second *largest real solution of the following* quartic *equation with respect to $t$:*

$$f_h(t) := t^4 + \xi_3 t^3 + \xi_2 t^2 + \xi_1 t + \xi_0 = 0, \tag{14}$$

*where the coefficients are defined by*

$$\xi_3 = \frac{(L-M)^2\gamma_h}{LM}, \quad \xi_2 = -\left(\xi_3\gamma_h + \frac{(L^2+M^2)\eta_h^2}{LM} + \frac{2\sigma^4}{c_{a_h}^2 c_{b_h}^2}\right), \quad \xi_1 = \xi_3\sqrt{\xi_0},$$

$$\xi_0 = \left(\eta_h^2 - \frac{\sigma^4}{c_{a_h}^2 c_{b_h}^2}\right)^2, \quad \eta_h^2 = \left(1 - \frac{\sigma^2 L}{\gamma_h^2}\right)\left(1 - \frac{\sigma^2 M}{\gamma_h^2}\right)\gamma_h^2.$$

*Let*

$$\widetilde{\gamma}_h = \sqrt{\frac{(L+M)\sigma^2}{2} + \frac{\sigma^4}{2c_{a_h}^2 c_{b_h}^2} + \sqrt{\left(\frac{(L+M)\sigma^2}{2} + \frac{\sigma^4}{2c_{a_h}^2 c_{b_h}^2}\right)^2 - LM\sigma^4}}. \tag{15}$$

*Then, the global VB solution* under matrix-wise independence (5) *can be expressed as*

$$\widehat{U}^{\mathrm{VB}} \equiv \langle BA^\top\rangle_{r^{\mathrm{VB}}(A,B)} = \widehat{B}\widehat{A}^\top = \sum_{h=1}^{H}\widehat{\gamma}_h^{\mathrm{VB}}\boldsymbol{\omega}_{b_h}\boldsymbol{\omega}_{a_h}^\top, \quad \text{where} \quad \widehat{\gamma}_h^{\mathrm{VB}} = \begin{cases}\widehat{\gamma}_h & \text{if } \gamma_h > \widetilde{\gamma}_h, \\ 0 & \text{otherwise.}\end{cases}$$

Theorem 1 holds also in the *empirical* Bayesian scenario, where the hyperparameters $(C_A, C_B)$ are also estimated from observation. Accordingly, the *empirical* VB solution also agrees with the *empirical* simpleVB solution, whose analytic-form is given in Corollary 5 in [15]. Thus, we obtain the global analytic solution for *empirical* VB:

**Corollary 2** *The global empirical VB solution* under matrix-wise independence (5) *is given by*

$$\widehat{U}^{\mathrm{EVB}} = \sum_{h=1}^{H}\widehat{\gamma}_h^{\mathrm{EVB}}\boldsymbol{\omega}_{b_h}\boldsymbol{\omega}_{a_h}^\top, \quad \text{where} \quad \widehat{\gamma}_h^{\mathrm{EVB}} = \begin{cases}\breve{\gamma}_h^{\mathrm{VB}} & \text{if } \gamma_h > \underline{\gamma}_h \text{ and } \Delta_h \leq 0, \\ 0 & \text{otherwise.}\end{cases}$$

*Here,*

$$\underline{\gamma}_h = (\sqrt{L} + \sqrt{M})\sigma, \tag{16}$$

$$\breve{c}_h^2 = \frac{1}{2LM}\left(\gamma_h^2 - (L+M)\sigma^2 + \sqrt{(\gamma_h^2 - (L+M)\sigma^2)^2 - 4LM\sigma^4}\right), \tag{17}$$

$$\Delta_h = M\log\left(\frac{\gamma_h}{M\sigma^2}\breve{\gamma}_h^{\mathrm{VB}} + 1\right) + L\log\left(\frac{\gamma_h}{L\sigma^2}\breve{\gamma}_h^{\mathrm{VB}} + 1\right) + \frac{1}{\sigma^2}\left(-2\gamma_h\breve{\gamma}_h^{\mathrm{VB}} + LM\breve{c}_h^2\right), \tag{18}$$

*and $\breve{\gamma}_h^{\mathrm{VB}}$ is the VB solution for $c_{a_h}c_{b_h} = \breve{c}_h$.*

When we calculate the *empirical* VB solution, we first check if $\gamma_h > \underline{\gamma}_h$ holds. If it holds, we compute $\breve{\gamma}_h^{\mathrm{VB}}$ by using Eq.(17) and Corollary 1. Otherwise, $\widehat{\gamma}_h^{\mathrm{EVB}} = 0$. Finally, we check if $\Delta_h \leq 0$ holds by using Eq.(18).

When the noise variance $\sigma^2$ is unknown, it is optimized by a naive 1-dimensional search to minimize the free energy [15]. To evaluate the free energy (13), we need the covariances $\Sigma_A$ and $\Sigma_B$, which neither Corollary 1 nor Corollary 2 provides. The following corollary, which gives the complete information on the VB posterior, is obtained by combining Theorem 1 above with Corollary 2 in [15]:

**Corollary 3** *The VB posteriors* under matrix-wise independence (5) *are given by*

$$r_{\mathrm{A}}^{\mathrm{VB}}(A) = \prod_{h=1}^{H} \mathcal{N}_M(\boldsymbol{a}_h; \widehat{\boldsymbol{a}}_h, \sigma_{a_h}^2 I_M), \quad r_{\mathrm{B}}^{\mathrm{VB}}(B) = \prod_{h=1}^{H} \mathcal{N}_L(\boldsymbol{b}_h; \widehat{\boldsymbol{b}}_h, \sigma_{b_h}^2 I_L),$$

*where, for $\widehat{\gamma}_h^{\mathrm{VB}}$ being the solution given by Corollary 1,*

$$\widehat{\boldsymbol{a}}_h = \pm\sqrt{\widehat{\gamma}_h^{\mathrm{VB}}\widehat{\delta}_h} \cdot \boldsymbol{\omega}_{a_h}, \quad \widehat{\boldsymbol{b}}_h = \pm\sqrt{\widehat{\gamma}_h^{\mathrm{VB}}\widehat{\delta}_h^{-1}} \cdot \boldsymbol{\omega}_{b_h},$$

$$\sigma_{a_h}^2 = \frac{-\left(\widehat{\eta}_h^2 - \sigma^2(M-L)\right) + \sqrt{(\widehat{\eta}_h^2 - \sigma^2(M-L))^2 + 4M\sigma^2\widehat{\eta}_h^2}}{2M(\widehat{\gamma}_h^{\mathrm{VB}}\widehat{\delta}_h^{-1} + \sigma^2 c_{a_h}^{-2})},$$

$$\sigma_{b_h}^2 = \frac{-\left(\widehat{\eta}_h^2 + \sigma^2(M-L)\right) + \sqrt{(\widehat{\eta}_h^2 + \sigma^2(M-L))^2 + 4L\sigma^2\widehat{\eta}_h^2}}{2L(\widehat{\gamma}_h^{\mathrm{VB}}\widehat{\delta}_h + \sigma^2 c_{b_h}^{-2})},$$

$$\widehat{\delta}_h = \frac{(M-L)(\gamma_h - \widehat{\gamma}_h^{\mathrm{VB}}) + \sqrt{(M-L)^2(\gamma_h - \widehat{\gamma}_h^{\mathrm{VB}})^2 + \frac{4\sigma^4 LM}{c_{a_h}^2 c_{b_h}^2}}}{2\sigma^2 M c_{a_h}^{-2}},$$

$$\widehat{\eta}_h^2 = \begin{cases} \eta_h^2 & \textit{if } \gamma_h > \widetilde{\gamma}_h, \\ \frac{\sigma^4}{c_{a_h}^2 c_{b_h}^2} & \textit{otherwise.} \end{cases}$$

Note that the ratio $c_{a_h}/c_{b_h}$ is arbitrary in empirical VB, so we can fix it to, e.g., $c_{a_h}/c_{b_h} = 1$ without loss of generality [15].

## 4 Experimental Results

In this section, we first introduce probabilistic PCA as a probabilistic MF model. Then, we show experimental results on artificial and benchmark datasets, which illustrate practical advantages of using our analytic solution.

### 4.1 Probabilistic PCA

In probabilistic PCA [20], the observation $\boldsymbol{y} \in \mathbb{R}^L$ is assumed to be driven by a latent vector $\widetilde{\boldsymbol{a}} \in \mathbb{R}^H$ in the following form:

$$\boldsymbol{y} = B\widetilde{\boldsymbol{a}} + \boldsymbol{\varepsilon}.$$

Here, $B \in \mathbb{R}^{L \times H}$ specifies the linear relationship between $\widetilde{\boldsymbol{a}}$ and $\boldsymbol{y}$, and $\boldsymbol{\varepsilon} \in \mathbb{R}^L$ is a Gaussian noise subject to $\mathcal{N}_L(\boldsymbol{0}, \sigma^2 I_L)$. Suppose that we are given $M$ observed samples $\{\boldsymbol{y}_1, \ldots, \boldsymbol{y}_M\}$ generated from the latent vectors $\{\widetilde{\boldsymbol{a}}_1, \ldots, \widetilde{\boldsymbol{a}}_M\}$, and each latent vector is subject to $\widetilde{\boldsymbol{a}} \sim \mathcal{N}_H(\boldsymbol{0}, I_H)$. Then, the probabilistic PCA model is written as Eqs.(1) and (2) with $C_A = I_H$.

If we apply Bayesian inference, the intrinsic dimension $H$ is automatically selected without predetermination [4, 14]. This useful property is called *automatic dimensionality selection* (ADS).

### 4.2 Experiment on Artificial Data

We compare the *iterative* algorithm and the *analytic* solution in the *empirical* VB scenario with unknown noise variance, i.e., the hyperparameters $(C_A, C_B)$ and the noise variance $\sigma^2$ are also

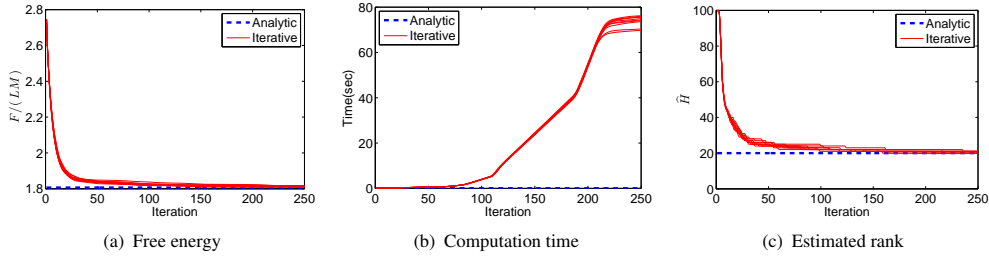

(a) Free energy        (b) Computation time        (c) Estimated rank

Figure 1: Experimental results for *Artificial1* dataset, where the data dimension is $L = 100$, the number of samples is $M = 300$, and the true rank is $H^* = 20$.

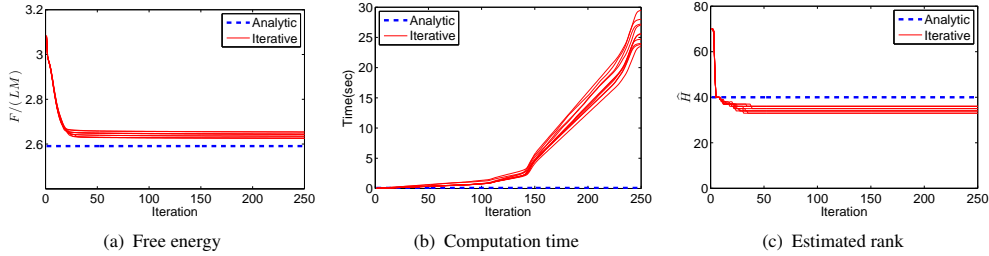

(a) Free energy        (b) Computation time        (c) Estimated rank

Figure 2: Experimental results for *Artificial2* dataset ($L = 70$, $M = 300$, and $H^* = 40$).

estimated from observation. We use the full-rank model (i.e., $H = \min(L, M)$), and expect the ADS effect to automatically find the true rank $H^*$.

Figure 1 shows the free energy, the computation time, and the estimated rank over iterations for an artificial (*Artificial1*) dataset with $L = 100$, $M = 300$, and $H^* = 20$. We randomly created *true* matrices $A^* \in \mathbb{R}^{M \times H^*}$ and $B^* \in \mathbb{R}^{L \times H^*}$ so that each entry of $A^*$ and $B^*$ follows $\mathcal{N}_1(0, 1)$. An observed matrix $Y$ was created by adding a noise subject to $\mathcal{N}_1(0, 1)$ to each entry of $B^* A^{*\top}$.

The iterative algorithm consists of the update rules (6)–(9). Initial values were set in the following way: $\widehat{A}$ and $\widehat{B}$ are randomly created so that each entry follows $\mathcal{N}_1(0, 1)$. Other variables are set to $\Sigma_A = \Sigma_B = C_A = C_B = I_H$ and $\sigma^2 = 1$. Note that we rescale $Y$ so that $\|Y\|_{\mathrm{Fro}}^2/(LM) = 1$, before starting iteration. We ran the iterative algorithm 10 times, starting from different initial points, and each trial is plotted by a solid line in Figure 1. The analytic solution consists of applying Corollary 2 combined with a naive 1-dimensional search for noise variance $\sigma^2$ estimation [15]. The analytic solution is plotted by the dashed line. We see that the analytic solution estimates the true rank $\widehat{H} = H^* = 20$ immediately ($\sim 0.1$ sec on average over 10 trials), while the iterative algorithm does not converge in 60 sec.

Figure 2 shows experimental results on another artificial dataset (*Artificial2*) where $L = 70$, $M = 300$, and $H^* = 40$. In this case, all the 10 trials of the iterative algorithm are trapped at local minima. We empirically observed a tendency that the iterative algorithm suffers from the local minima problem when $H^*$ is large (close to $H$).

## 4.3 Experiment on Benchmark Data

Figures 3 and 4 show experimental results on the *Satellite* and the *Spectf* datasets available from the UCI repository [1], showing similar tendencies to Figures 1 and 2. We also conducted experiments on various benchmark datasets, and found that the iterative algorithm typically converges slowly, and sometimes suffers from the local minima problem, while our analytic-form gives the global solution immediately.

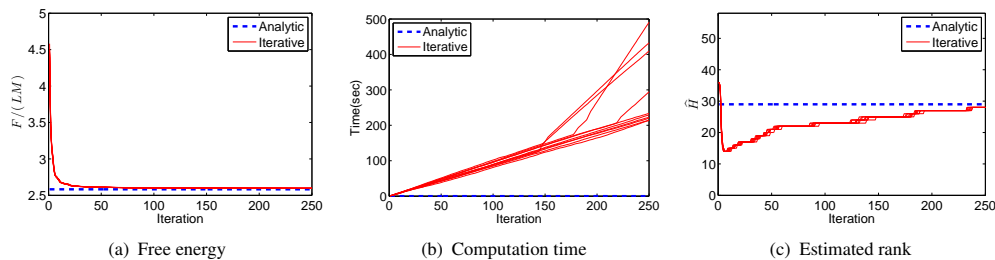

(a) Free energy       (b) Computation time       (c) Estimated rank

Figure 3: Experimental results for the *Sat* dataset ($L = 36, M = 6435$).

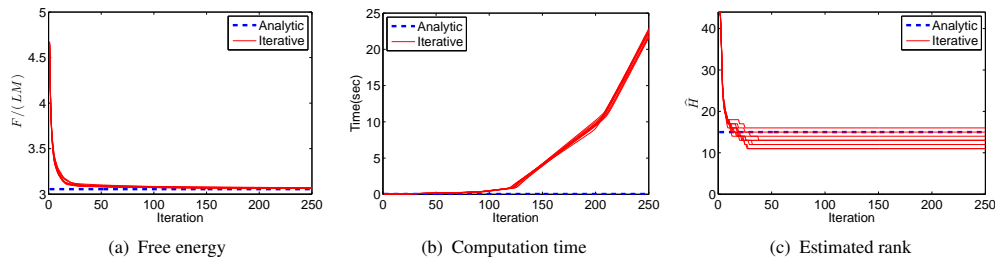

(a) Free energy       (b) Computation time       (c) Estimated rank

Figure 4: Experimental results for the *Spectf* dataset ($L = 44, M = 267$).

## 5   Conclusion and Discussion

In this paper, we have analyzed the fully-observed variational Bayesian matrix factorization (VBMF) under matrix-wise independence. We have shown that the VB solution under matrix-wise independence essentially agrees with the simplified VB (simpleVB) solution under *column-wise* independence. As a consequence, we can obtain the global VB solution under matrix-wise independence *analytically* in a computationally very efficient way.

Our analysis assumed uncorrelated priors. With correlated priors, the posterior is no longer uncorrelated and thus it is not straightforward to obtain a global solution analytically. Nevertheless, there exists a situation where an analytic solution can be easily obtained: Suppose there exists an $H \times H$ non-singular matrix $T$ such that both of $C'_A = TC_AT^\top$ and $C'_B = (T^{-1})^\top C_BT^{-1}$ are diagonal. We can show that the free energy (13) is invariant under the following transformation for any $T$:

$$A \to AT^\top, \quad\quad \Sigma_A \to T\Sigma_AT^\top, \quad\quad C_A \to TC_AT^\top,$$
$$B \to BT^{-1}, \quad\quad \Sigma_B \to (T^{-1})^T\Sigma_BT^{-1}, \quad\quad C_B \to (T^{-1})^\top C_BT^{-1}.$$

Accordingly, the following procedure gives the global solution analytically: the analytic solution given the diagonal $(C'_A, C'_B)$ is first computed, and the above transformation is then applied.

We have demonstrated the usefulness of our analytic solution in probabilistic PCA. On the other hand, robust PCA has gathered a great deal of attention recently [5], and its Bayesian variant has been proposed [2]. We expect that our analysis can handle more structured sparsity, in addition to the current low-rank inducing sparsity. Extension of the current work along this line will allow us to give more theoretical insights into robust PCA and provide computationally efficient algorithms.

Finally, a more challenging direction is to handle priors correlated over *rows* of $A$ and $B$. This allows us to model correlations in the observation space, and capture, e.g., short-term correlation in time-series data and neighboring pixels correlation in image data. Analyzing such a situation, as well as missing value imputation and tensor factorization [11, 6, 8, 21] is our important future work.

### Acknowledgments

The authors thank anonymous reviewers for helpful comments. Masashi Sugiyama was supported by the FIRST program. Derin Babacan was supported by a Beckman Postdoctoral Fellowship.

## Footnotes

[1]This excludes the *collaborative filtering* setup, which is aimed at imputing missing entries of an observed matrix [12, 7].

# References

[1] A. Asuncion and D.J. Newman. UCI machine learning repository, 2007.

[2] D. Babacan, M. Luessi, R. Molina, and A. Katsaggelos. Sparse Bayesian methods for low-rank matrix estimation. *arXiv:1102.5288v1 [stat.ML]*, 2011.

[3] C. M. Bishop. Bayesian principal components. In *Advances in NIPS*, volume 11, pages 382–388, 1999.

[4] C. M. Bishop. Variational principal components. In *Proc. of ICANN*, volume 1, pages 514–509, 1999.

[5] E.-J. Candes, X. Li, Y. Ma, and J. Wright. Robust principal component analysis? *CoRR*, abs/0912.3599, 2009.

[6] J. D. Carroll and J. J. Chang. Analysis of individual differences in multidimensional scaling via an n-way generalization of 'eckart-young' decomposition. *Psychometrika*, 35:283–319, 1970.

[7] S. Funk. Try this at home. http://sifter.org/~simon/journal/20061211.html, 2006.

[8] R. A. Harshman. Foundations of the parafac procedure: Models and conditions for an "explanatory" multimodal factor analysis. *UCLA Working Papers in Phonetics*, 16:1–84, 1970.

[9] H. Hotelling. Analysis of a complex of statistical variables into principal components. *Journal of Educational Psychology*, 24:417–441, 1933.

[10] H. Hotelling. Relations between two sets of variates. *Biometrika*, 28(3–4):321–377, 1936.

[11] T. G. Kolda and B. W. Bader. Tensor decompositions and applications. *SIAM Review*, 51(3):455–500, 2009.

[12] J. A. Konstan, B. N. Miller, D. Maltz, J. L. Herlocker, L. R. Gordon, and J. Riedl. Grouplens: Applying collaborative filtering to Usenet news. *Communications of the ACM*, 40(3):77–87, 1997.

[13] Y. J. Lim and T. W. Teh. Variational Bayesian approach to movie rating prediction. In *Proceedings of KDD Cup and Workshop*, 2007.

[14] S. Nakajima, M. Sugiyama, and D. Babacan. On Bayesian PCA: Automatic dimensionality selection and analytic solution. In *Proceedings of 28th International Conference on Machine Learning (ICML2011)*, Bellevue, WA, USA, Jun. 28–Jul.2 2011.

[15] S. Nakajima, M. Sugiyama, and R. Tomioka. Global analytic solution for variational Bayesian matrix factorization. In J. Lafferty, C. K. I. Williams, R. Zemel, J. Shawe-Taylor, and A. Culotta, editors, *Advances in Neural Information Processing Systems 23*, pages 1759–1767, 2010.

[16] T. Raiko, A. Ilin, and J. Karhunen. Principal component analysis for large scale problems with lots of missing values. In J. Kok, J. Koronacki, R. Lopez de Mantras, S. Matwin, D. Mladenic, and A. Skowron, editors, *Proceedings of the 18th European Conference on Machine Learning*, volume 4701 of *Lecture Notes in Computer Science*, pages 691–698, Berlin, 2007. Springer-Verlag.

[17] S. Roweis and Z. Ghahramani. A unifying review of linear Gaussian models. *Neural Computation*, 11:305–345, 1999.

[18] R. Salakhutdinov and A. Mnih. Bayesian probabilistic matrix factorization using Markov chain Monte Carlo. In *International Conference on Machine Learning*, 2008.

[19] R. Salakhutdinov and A. Mnih. Probabilistic matrix factorization. In J. C. Platt, D. Koller, Y. Singer, and S. Roweis, editors, *Advances in Neural Information Processing Systems 20*, pages 1257–1264, Cambridge, MA, 2008. MIT Press.

[20] M. E. Tipping and C. M. Bishop. Probabilistic principal component analysis. *Journal of the Royal Statistical Society*, 61:611–622, 1999.

[21] L. R. Tucker. Some mathematical notes on three-mode factor analysis. *Psychometrika*, 31:279–311, 1996.

